# Policy Gradient Methods for Reinforcement Learning with Function Approximation

Richard S. Sutton, David McAllester, Satinder Singh, Yishay Mansour
AT&T Labs – Research, 180 Park Avenue, Florham Park, NJ 07932

## Abstract

Function approximation is essential to reinforcement learning, but the standard approach of approximating a value function and determining a policy from it has so far proven theoretically intractable. In this paper we explore an alternative approach in which the policy is explicitly represented by its own function approximator, independent of the value function, and is updated according to the gradient of expected reward with respect to the policy parameters. Williams's REINFORCE method and actor–critic methods are examples of this approach. Our main new result is to show that the gradient can be written in a form suitable for estimation from experience aided by an approximate action-value or advantage function. Using this result, we prove for the first time that a version of policy iteration with arbitrary differentiable function approximation is convergent to a locally optimal policy.

Large applications of reinforcement learning (RL) require the use of generalizing function approximators such neural networks, decision-trees, or instance-based methods. The dominant approach for the last decade has been the *value-function* approach, in which all function approximation effort goes into estimating a value function, with the action-selection policy represented implicitly as the "greedy" policy with respect to the estimated values (e.g., as the policy that selects in each state the action with highest estimated value). The value-function approach has worked well in many applications, but has several limitations. First, it is oriented toward finding deterministic policies, whereas the optimal policy is often stochastic, selecting different actions with specific probabilities (e.g., see Singh, Jaakkola, and Jordan, 1994). Second, an arbitrarily small change in the estimated value of an action can cause it to be, or not be, selected. Such discontinuous changes have been identified as a key obstacle to establishing convergence assurances for algorithms following the value-function approach (Bertsekas and Tsitsiklis, 1996). For example, Q-learning, Sarsa, and dynamic programming methods have all been shown unable to converge to any policy for simple MDPs and simple function approximators (Gordon, 1995, 1996; Baird, 1995; Tsitsiklis and van Roy, 1996; Bertsekas and Tsitsiklis, 1996). This can occur even if the best approximation is found at each step before changing the policy, and whether the notion of "best" is in the mean-squared-error sense or the slightly different senses of residual-gradient, temporal-difference, and dynamic-programming methods.

In this paper we explore an alternative approach to function approximation in RL.

Rather than approximating a value function and using that to compute a deterministic policy, we approximate a stochastic policy directly using an independent function approximator with its own parameters. For example, the policy might be represented by a neural network whose input is a representation of the state, whose output is action selection probabilities, and whose weights are the policy parameters. Let $\theta$ denote the vector of policy parameters and $\rho$ the performance of the corresponding policy (e.g., the average reward per step). Then, in the *policy gradient* approach, the policy parameters are updated approximately proportional to the gradient:

$$\Delta\theta \approx \alpha\frac{\partial\rho}{\partial\theta}, \tag{1}$$

where $\alpha$ is a positive-definite step size. If the above can be achieved, then $\theta$ can usually be assured to converge to a locally optimal policy in the performance measure $\rho$. Unlike the value-function approach, here small changes in $\theta$ can cause only small changes in the policy and in the state-visitation distribution.

In this paper we prove that an unbiased estimate of the gradient (1) can be obtained from experience using an approximate value function satisfying certain properties. Williams's (1988, 1992) REINFORCE algorithm also finds an unbiased estimate of the gradient, but without the assistance of a learned value function. REINFORCE learns much more slowly than RL methods using value functions and has received relatively little attention. Learning a value function and using it to reduce the variance of the gradient estimate appears to be essential for rapid learning. Jaakkola, Singh and Jordan (1995) proved a result very similar to ours for the special case of function approximation corresponding to tabular POMDPs. Our result strengthens theirs and generalizes it to arbitrary differentiable function approximators. Konda and Tsitsiklis (in prep.) independently developed a very simialr result to ours. See also Baxter and Bartlett (in prep.) and Marbach and Tsitsiklis (1998).

Our result also suggests a way of proving the convergence of a wide variety of algorithms based on "actor-critic" or policy-iteration architectures (e.g., Barto, Sutton, and Anderson, 1983; Sutton, 1984; Kimura and Kobayashi, 1998). In this paper we take the first step in this direction by proving for the first time that a version of policy iteration with general differentiable function approximation is convergent to a locally optimal policy. Baird and Moore (1999) obtained a weaker but superficially similar result for their VAPS family of methods. Like policy-gradient methods, VAPS includes separately parameterized policy and value functions updated by gradient methods. However, VAPS methods do not climb the gradient of performance (expected long-term reward), but of a measure combining performance and value-function accuracy. As a result, VAPS does not converge to a locally optimal policy, except in the case that no weight is put upon value-function accuracy, in which case VAPS degenerates to REINFORCE. Similarly, Gordon's (1995) fitted value iteration is also convergent and value-based, but does not find a locally optimal policy.

# 1 Policy Gradient Theorem

We consider the standard reinforcement learning framework (see, e.g., Sutton and Barto, 1998), in which a learning agent interacts with a Markov decision process (MDP). The state, action, and reward at each time $t \in \{0, 1, 2, \ldots\}$ are denoted $s_t \in \mathcal{S}$, $a_t \in \mathcal{A}$, and $r_t \in \Re$ respectively. The environment's dynamics are characterized by state transition probabilities, $\mathcal{P}_{ss'}^a = Pr\{s_{t+1} = s' \mid s_t = s, a_t = a\}$, and expected rewards $\mathcal{R}_s^a = E\{r_{t+1} \mid s_t = s, a_t = a\}$, $\forall s, s' \in \mathcal{S}, a \in \mathcal{A}$. The agent's decision making procedure at each time is characterized by a policy, $\pi(s, a, \theta) = Pr\{a_t = a | s_t = s, \theta\}$, $\forall s \in \mathcal{S}, a \in \mathcal{A}$, where $\theta \in \Re^l$, for $l << |\mathcal{S}|$, is a parameter vector. We assume that $\pi$ is diffentiable with respect to its parameter, i.e., that $\frac{\partial\pi(s,a)}{\partial\theta}$ exists. We also usually write just $\pi(s, a)$ for $\pi(s, a, \theta)$.

With function approximation, two ways of formulating the agent's objective are useful. One is the average reward formulation, in which policies are ranked according to their long-term expected reward per step, $\rho(\pi)$:

$$\rho(\pi) = \lim_{n \to \infty} \frac{1}{n} E\{r_1 + r_2 + \cdots + r_n \mid \pi\} = \sum_s d^\pi(s) \sum_a \pi(s,a) \mathcal{R}_s^a,$$

where $d^\pi(s) = \lim_{t \to \infty} Pr\{s_t = s|s_0, \pi\}$ is the stationary distribution of states under $\pi$, which we assume exists and is independent of $s_0$ for all policies. In the average reward formulation, the value of a state–action pair given a policy is defined as

$$Q^\pi(s,a) = \sum_{t=1}^\infty E\{r_t - \rho(\pi) \mid s_0 = s, a_0 = a, \pi\}, \qquad \forall s \in \mathcal{S}, a \in \mathcal{A}.$$

The second formulation we cover is that in which there is a designated start state $s_0$, and we care only about the long-term reward obtained from it. We will give our results only once, but they will apply to this formulation as well under the definitions

$$\rho(\pi) = E\left\{\sum_{t=1}^\infty \gamma^{t-1} r_t \;\middle|\; s_0, \pi\right\} \quad \text{and} \quad Q^\pi(s,a) = E\left\{\sum_{k=1}^\infty \gamma^{k-1} r_{t+k} \;\middle|\; s_t = s, a_t = a, \pi\right\}.$$

where $\gamma \in [0,1]$ is a discount rate ($\gamma = 1$ is allowed only in episodic tasks). In this formulation, we define $d^\pi(s)$ as a discounted weighting of states encountered starting at $s_0$ and then following $\pi$: $d^\pi(s) = \sum_{t=0}^\infty \gamma^t Pr\{s_t = s|s_0, \pi\}$.

Our first result concerns the gradient of the performance metric with respect to the policy parameter:

**Theorem 1 (Policy Gradient).** For any MDP, in either the average-reward or start-state formulations,

$$\frac{\partial \rho}{\partial \theta} = \sum_s d^\pi(s) \sum_a \frac{\partial \pi(s,a)}{\partial \theta} Q^\pi(s,a). \tag{2}$$

**Proof:** See the appendix.

This way of expressing the gradient was first discussed for the average-reward formulation by Marbach and Tsitsiklis (1998), based on a related expression in terms of the state-value function due to Jaakkola, Singh, and Jordan (1995) and Cao and Chen (1997). We extend their results to the start-state formulation and provide simpler and more direct proofs. Williams's (1988, 1992) theory of REINFORCE algorithms can also be viewed as implying (2). In any event, the key aspect of both expressions for the gradient is that their are no terms of the form $\frac{\partial d^\pi(s)}{\partial \theta}$: the effect of policy changes on the distribution of states does not appear. This is convenient for approximating the gradient by sampling. For example, if $s$ was sampled from the distribution obtained by following $\pi$, then $\sum_a \frac{\partial \pi(s,a)}{\partial \theta} Q^\pi(s,a)$ would be an unbiased estimate of $\frac{\partial \rho}{\partial \theta}$. Of course, $Q^\pi(s,a)$ is also not normally known and must be estimated. One approach is to use the actual returns, $R_t = \sum_{k=1}^\infty r_{t+k} - \rho(\pi)$ (or $R_t = \sum_{k=1}^\infty \gamma^{k-1} r_{t+k}$ in the start-state formulation) as an approximation for each $Q^\pi(s_t, a_t)$. This leads to Williams's episodic REINFORCE algorithm, $\Delta\theta_t \propto \frac{\partial \pi(s_t, a_t)}{\partial \theta} R_t \frac{1}{\pi(s_t, a_t)}$ (the $\frac{1}{\pi(s_t, a_t)}$ corrects for the oversampling of actions preferred by $\pi$), which is known to follow $\frac{\partial \rho}{\partial \theta}$ in expected value (Williams, 1988, 1992).

## 2  Policy Gradient with Approximation

Now consider the case in which $Q^\pi$ is approximated by a learned function approximator. If the approximation is sufficiently good, we might hope to use it in place of $Q^\pi$

in (2) and still point roughly in the direction of the gradient. For example, Jaakkola, Singh, and Jordan (1995) proved that for the special case of function approximation arising in a tabular POMDP one could assure positive inner product with the gradient, which is sufficient to ensure improvement for moving in that direction. Here we extend their result to general function approximation and prove equality with the gradient.

Let $f_w : \mathcal{S} \times \mathcal{A} \rightarrow \Re$ be our approximation to $Q^\pi$, with parameter $w$. It is natural to learn $f_w$ by following $\pi$ and updating $w$ by a rule such as $\Delta w_t \propto \frac{\partial}{\partial w} [\hat{Q}^\pi(s_t, a_t) - f_w(s_t, a_t)]^2 \propto [\hat{Q}^\pi(s_t, a_t) - f_w(s_t, a_t)] \frac{\partial f_w(s_t, a_t)}{\partial w}$, where $\hat{Q}^\pi(s_t, a_t)$ is some unbiased estimator of $Q^\pi(s_t, a_t)$, perhaps $R_t$. When such a process has converged to a local optimum, then

$$\sum_s d^\pi(s) \sum_a \pi(s, a) [Q^\pi(s, a) - f_w(s, a)] \frac{\partial f_w(s, a)}{\partial w} = 0. \qquad (3)$$

**Theorem 2 (Policy Gradient with Function Approximation).** If $f_w$ satisfies (3) and is compatible with the policy parameterization in the sense that[1]

$$\frac{\partial f_w(s, a)}{\partial w} = \frac{\partial \pi(s, a)}{\partial \theta} \frac{1}{\pi(s, a)}, \qquad (4)$$

then

$$\frac{\partial \rho}{\partial \theta} = \sum_s d^\pi(s) \sum_a \frac{\partial \pi(s, a)}{\partial \theta} f_w(s, a). \qquad (5)$$

**Proof:** Combining (3) and (4) gives

$$\sum_s d^\pi(s) \sum_a \frac{\partial \pi(s, a)}{\partial \theta} [Q^\pi(s, a) - f_w(s, a)] = 0 \qquad (6)$$

which tells us that the error in $f_w(s, a)$ is orthogonal to the gradient of the policy parameterization. Because the expression above is zero, we can subtract it from the policy gradient theorem (2) to yield

$$\begin{aligned}
\frac{\partial \rho}{\partial \theta} &= \sum_s d^\pi(s) \sum_a \frac{\partial \pi(s, a)}{\partial \theta} Q^\pi(s, a) - \sum_s d^\pi(s) \sum_a \frac{\partial \pi(s, a)}{\partial \theta} [Q^\pi(s, a) - f_w(s, a)] \\
&= \sum_s d^\pi(s) \sum_a \frac{\partial \pi(s, a)}{\partial \theta} [Q^\pi(s, a) - Q^\pi(s, a) + f_w(s, a)] \\
&= \sum_s d^\pi(s) \sum_a \frac{\partial \pi(s, a)}{\partial \theta} f_w(s, a). \qquad \text{Q.E.D.}
\end{aligned}$$

## 3   Application to Deriving Algorithms and Advantages

Given a policy parameterization, Theorem 2 can be used to derive an appropriate form for the value-function parameterization. For example, consider a policy that is a Gibbs distribution in a linear combination of features:

$$\pi(s, a) = \frac{e^{\theta^T \phi_{sa}}}{\sum_b e^{\theta^T \phi_{sb}}}, \qquad \forall s \in \mathcal{S}, s \in \mathcal{A},$$

where each $\phi_{sa}$ is an $l$-dimensional feature vector characterizing state-action pair $s, a$. Meeting the compatibility condition (4) requires that

$$\frac{\partial f_w(s,a)}{\partial w} = \frac{\partial \pi(s,a)}{\partial \theta} \frac{1}{\pi(s,a)} = \phi_{sa} - \sum_b \pi(s,b)\phi_{sb},$$

so that the natural parameterization of $f_w$ is

$$f_w(s,a) = w^T \left[ \phi_{sa} - \sum_b \pi(s,b)\phi_{sb} \right].$$

In other words, $f_w$ must be linear in the same features as the policy, except normalized to be mean zero for each state. Other algorithms can easily be derived for a variety of nonlinear policy parameterizations, such as multi-layer backpropagation networks.

The careful reader will have noticed that the form given above for $f_w$ requires that it have zero mean for each state: $\sum_a \pi(s,a)f_w(s,a) = 0$, $\forall s \in \mathcal{S}$. In this sense it is better to think of $f_w$ as an approximation of the *advantage* function, $A^\pi(s,a) = Q^\pi(s,a) - V^\pi(s)$ (much as in Baird, 1993), rather than of $Q^\pi$. Our convergence requirement (3) is really that $f_w$ get the relative value of the actions correct in each state, not the absolute value, nor the variation from state to state. Our results can be viewed as a justification for the special status of advantages as the target for value function approximation in RL. In fact, our (2), (3), and (5), can all be generalized to include an arbitrary function of state added to the value function or its approximation. For example, (5) can be generalized to $\frac{\partial \rho}{\partial \theta} = \sum_s d^\pi(s) \sum_a \frac{\partial \pi(s,a)}{\partial \theta} [f_w(s,a) + v(s)]$, where $v : \mathcal{S} \to \Re$ is an arbitrary function. (This follows immediately because $\sum_a \frac{\partial \pi(s,a)}{\partial \theta} = 0$, $\forall s \in \mathcal{S}$.) The choice of $v$ does not affect any of our theorems, but can substantially affect the variance of the gradient estimators. The issues here are entirely analogous to those in the use of reinforcement baselines in earlier work (e.g., Williams, 1992; Dayan, 1991; Sutton, 1984). In practice, $v$ should presumably be set to the best available approximation of $V^\pi$. Our results establish that that approximation process can proceed without affecting the expected evolution of $f_w$ and $\pi$.

## 4 Convergence of Policy Iteration with Function Approximation

Given Theorem 2, we can prove for the first time that a form of policy iteration with function approximation is convergent to a locally optimal policy.

**Theorem 3 (Policy Iteration with Function Approximation).** Let $\pi$ and $f_w$ be any differentiable function approximators for the policy and value function respectively that satisfy the compatibility condition (4) and for which $\max_{\theta,s,a,i,j} |\frac{\partial^2 \pi(s,a)}{\partial \theta_i \partial \theta_j}| < B < \infty$. Let $\{\alpha_k\}_{k=0}^\infty$ be any step-size sequence such that $\lim_{k \to \infty} \alpha_k = 0$ and $\sum_k \alpha_k = \infty$. Then, for any MDP with bounded rewards, the sequence $\{\rho(\pi_k)\}_{k=0}^\infty$, defined by any $\theta_0$, $\pi_k = \pi(\cdot,\cdot,\theta_k)$, and

$$w_k = w \text{ such that } \sum_s d^{\pi_k}(s) \sum_a \pi_k(s,a)[Q^{\pi_k}(s,a) - f_w(s,a)]\frac{\partial f_w(s,a)}{\partial w} = 0$$

$$\theta_{k+1} = \theta_k + \alpha_k \sum_s d^{\pi_k}(s) \sum_a \frac{\partial \pi_k(s,a)}{\partial \theta} f_{w_k}(s,a),$$

converges such that $\lim_{k \to \infty} \frac{\partial \rho(\pi_k)}{\partial \theta} = 0$.

**Proof:** Our Theorem 2 assures that the $\theta_k$ update is in the direction of the gradient. The bounds on $\frac{\partial^2 \pi(s,a)}{\partial \theta_i \partial \theta_j}$ and on the MDP's rewards together assure us that $\frac{\partial^2 \rho}{\partial \theta_i \partial \theta_j}$

is also bounded. These, together with the step-size requirements, are the necessary conditions to apply Proposition 3.5 from page 96 of Bertsekas and Tsitsiklis (1996), which assures convergence to a local optimum.                                      Q.E.D.

## Acknowledgements

The authors wish to thank Martha Steenstrup and Doina Precup for comments, and Michael Kearns for insights into the notion of optimal policy under function approximation.

## Footnotes

[1]Tsitsiklis (personal communication) points out that $f_w$ being linear in the features given on the righthand side may be the only way to satisfy this condition.

## References

Baird, L. C. (1993). Advantage Updating. Wright Lab. Technical Report WL-TR-93-1146.

Baird, L. C. (1995). Residual algorithms: Reinforcement learning with function approximation. *Proc. of the Twelfth Int. Conf. on Machine Learning*, pp. 30–37. Morgan Kaufmann.

Baird, L. C., Moore, A. W. (1999). Gradient descent for general reinforcement learning. *NIPS 11*. MIT Press.

Barto, A. G., Sutton, R. S., Anderson, C. W. (1983). Neuronlike elements that can solve difficult learning control problems. *IEEE Trans. on Systems, Man, and Cybernetics 13*:835.

Baxter, J., Bartlett, P. (in prep.) Direct gradient-based reinforcement learning: I. Gradient estimation algorithms.

Bertsekas, D. P., Tsitsiklis, J. N. (1996). *Neuro-Dynamic Programming*. Athena Scientific.

Cao, X.-R., Chen, H.-F. (1997). Perturbation realization, potentials, and sensitivity analysis of Markov Processes, *IEEE Trans. on Automatic Control 42*(10):1382–1393.

Dayan, P. (1991). Reinforcement comparison. In D. S. Touretzky, J. L. Elman, T. J. Sejnowski, and G. E. Hinton (eds.), *Connectionist Models: Proceedings of the 1990 Summer School*, pp. 45–51. Morgan Kaufmann.

Gordon, G. J. (1995). Stable function approximation in dynamic programming. *Proceedings of the Twelfth Int. Conf. on Machine Learning*, pp. 261–268. Morgan Kaufmann.

Gordon, G. J. (1996). Chattering in SARSA($\lambda$). CMU Learning Lab Technical Report.

Jaakkola, T., Singh, S. P., Jordan, M. I. (1995) Reinforcement learning algorithms for partially observable Markov decision problems, *NIPS 7*, pp. 345–352. Morgan Kaufman.

Kimura, H., Kobayashi, S. (1998). An analysis of actor/critic algorithms using eligibility traces: Reinforcement learning with imperfect value functions. *Proc. ICML-98*, pp. 278-286.

Konda, V. R., Tsitsiklis, J. N. (in prep.) Actor-critic algorithms.

Marbach, P., Tsitsiklis, J. N. (1998) Simulation-based optimization of Markov reward processes, technical report LIDS-P-2411, Massachusetts Institute of Technology.

Singh, S. P., Jaakkola, T., Jordan, M. I. (1994). Learning without state-estimation in partially observable Markovian decision problems. *Proc. ICML-94*, pp. 284–292.

Sutton, R. S. (1984). *Temporal Credit Assignment in Reinforcement Learning*. Ph.D. thesis, University of Massachusetts, Amherst.

Sutton, R. S., Barto, A. G. (1998). *Reinforcement Learning: An Introduction*. MIT Press.

Tsitsiklis, J. N. Van Roy, B. (1996). Feature-based methods for large scale dynamic programming. *Machine Learning 22*:59–94.

Williams, R. J. (1988). Toward a theory of reinforcement-learning connectionist systems. Technical Report NU-CCS-88-3, Northeastern University, College of Computer Science.

Williams, R. J. (1992). Simple statistical gradient-following algorithms for connectionist reinforcement learning. *Machine Learning 8*:229–256.

## Appendix: Proof of Theorem 1

We prove the theorem first for the average-reward formulation and then for the start-state formulation.

$$\frac{\partial V^\pi(s)}{\partial \theta} \overset{\text{def}}{=} \frac{\partial}{\partial \theta} \sum_a \pi(s,a) Q^\pi(s,a) \qquad \forall s \in \mathcal{S}$$

$$= \sum_a \left[ \frac{\partial \pi(s,a)}{\partial \theta} Q^\pi(s,a) + \pi(s,a) \frac{\partial}{\partial \theta} Q^\pi(s,a) \right]$$

$$= \sum_a \left[ \frac{\partial \pi(s,a)}{\partial \theta} Q^\pi(s,a) + \pi(s,a) \frac{\partial}{\partial \theta} \left[ \mathcal{R}_s^a - \rho(\pi) + \sum_{s'} \mathcal{P}_{ss'}^a V^\pi(s') \right] \right]$$

$$= \sum_a \left[ \frac{\partial \pi(s,a)}{\partial \theta} Q^\pi(s,a) + \pi(s,a) \left[ -\frac{\partial \rho}{\partial \theta} + \sum_{s'} \mathcal{P}_{ss'}^a \frac{\partial V^\pi(s')}{\partial \theta} \right] \right]$$

Therefore,

$$\frac{\partial \rho}{\partial \theta} = \sum_a \left[ \frac{\partial \pi(s,a)}{\partial \theta} Q^\pi(s,a) + \pi(s,a) \sum_{s'} \mathcal{P}_{ss'}^a \frac{\partial V^\pi(s')}{\partial \theta} \right] - \frac{\partial V^\pi(s)}{\partial \theta}$$

Summing both sides over the stationary distribution $d^\pi$,

$$\sum_s d^\pi(s) \frac{\partial \rho}{\partial \theta} = \sum_s d^\pi(s) \sum_a \frac{\partial \pi(s,a)}{\partial \theta} Q^\pi(s,a) + \sum_s d^\pi(s) \sum_a \pi(s,a) \sum_{s'} \mathcal{P}_{ss'}^a \frac{\partial V^\pi(s')}{\partial \theta}$$

$$- \sum_s d^\pi(s) \frac{\partial V^\pi(s)}{\partial \theta},$$

but since $d^\pi$ is stationary,

$$\sum_s d^\pi(s) \frac{\partial \rho}{\partial \theta} = \sum_s d^\pi(s) \sum_a \frac{\partial \pi(s,a)}{\partial \theta} Q^\pi(s,a) + \sum_{s'} d^\pi(s') \frac{\partial V^\pi(s')}{\partial \theta}$$

$$- \sum_s d^\pi(s) \frac{\partial V^\pi(s)}{\partial \theta}$$

$$\frac{\partial \rho}{\partial \theta} = \sum_s d^\pi(s) \sum_a \frac{\partial \pi(s,a)}{\partial \theta} Q^\pi(s,a). \qquad \text{Q.E.D.}$$

For the start-state formulation:

$$\frac{\partial V^\pi(s)}{\partial \theta} \overset{\text{def}}{=} \frac{\partial}{\partial \theta} \sum_a \pi(s,a) Q^\pi(s,a) \qquad \forall s \in \mathcal{S}$$

$$= \sum_a \left[ \frac{\partial \pi(s,a)}{\partial \theta} Q^\pi(s,a) + \pi(s,a) \frac{\partial}{\partial \theta} Q^\pi(s,a) \right]$$

$$= \sum_a \left[ \frac{\partial \pi(s,a)}{\partial \theta} Q^\pi(s,a) + \pi(s,a) \frac{\partial}{\partial \theta} \left[ \mathcal{R}_s^a + \sum_{s'} \gamma \mathcal{P}_{ss'}^a V^\pi(s') \right] \right]$$

$$= \sum_a \left[ \frac{\partial \pi(s,a)}{\partial \theta} Q^\pi(s,a) + \pi(s,a) \sum_{s'} \gamma \mathcal{P}_{ss'}^a \frac{\partial}{\partial \theta} V^\pi(s') \right] \qquad (7)$$

$$= \sum_x \sum_{k=0}^\infty \gamma^k Pr(s \to x, k, \pi) \sum_a \frac{\partial \pi(x,a)}{\partial \theta} Q^\pi(x,a),$$

after several steps of unrolling (7), where $Pr(s \to x, k, \pi)$ is the probability of going from state $s$ to state $x$ in $k$ steps under policy $\pi$. It is then immediate that

$$\frac{\partial \rho}{\partial \theta} = \frac{\partial}{\partial \theta} E \left\{ \sum_{t=1}^\infty \gamma^{t-1} r_t \,\Big|\, s_0, \pi \right\} = \frac{\partial}{\partial \theta} V^\pi(s_0)$$

$$= \sum_s \sum_{k=0}^\infty \gamma^k Pr(s_0 \to s, k, \pi) \sum_a \frac{\partial \pi(s,a)}{\partial \theta} Q^\pi(s,a)$$

$$= \sum_s d^\pi(s) \sum_a \frac{\partial \pi(s,a)}{\partial \theta} Q^\pi(s,a). \qquad \text{Q.E.D.}$$